# Circuit Model of Short-Term Synaptic Dynamics

**Shih-Chii Liu, Malte Boegershausen, and Pascal Suter**
Institute of Neuroinformatics
University of Zurich and ETH Zurich
Winterthurerstrasse 190
CH-8057 Zurich, Switzerland
shih@ini.phys.ethz.ch

## Abstract

We describe a model of short-term synaptic depression that is derived from a silicon circuit implementation. The dynamics of this circuit model are similar to the dynamics of some present theoretical models of short-term depression except that the recovery dynamics of the variable describing the depression is nonlinear and it also depends on the presynaptic frequency. The equations describing the steady-state and transient responses of this synaptic model fit the experimental results obtained from a fabricated silicon network consisting of leaky integrate-and-fire neurons and different types of synapses. We also show experimental data demonstrating the possible computational roles of depression. One possible role of a depressing synapse is that the input can quickly bring the neuron up to threshold when the membrane potential is close to the resting potential.

## 1   Introduction

Short-term synaptic dynamics have been observed in many parts of the cortical system [Stratford et al., 1998, Varela et al., 1997, Tsodyks et al., 1998]. The functionality of the short-term synaptic dynamics have been implicated in various cortical models [Senn et al., 1998, Chance et al., 1998, Matveev and Wang, 2000]. along with the processing capabilities of a network with dynamic synapses [Tsodyks et al., 1998, Maass and Zador, 1999]. The introduction of these dynamic synapses into hardware implementations of recurrent neuronal networks allow a wide range of operating regimes especially in the case of time-varying inputs.

In this work, we describe a model that was derived from a circuit implementation of short-term depression. The circuit implementation was initially described by [Rasche and Hahnloser, 2001] but the dynamics were not analyzed in their work. We also compare the dynamics of the circuit model of depression with the equations of one of the theoretical models frequently used in network simulations [Abbott et al., 1997, Varela et al., 1997] and show examples of transient and steady-state responses of this synaptic circuit to inputs of different statistical distributions.

This circuit has been included in a silicon network of leaky integrate-and-fire neurons together with other short-term dynamic synapses like facilitation synapses. We also show

experimental data from the chip that demonstrate the possible computational roles of depression. We postulate that one possible role of depression is to bring the neuron's response quickly up to threshold if the membrane potential of the neuron was close to the resting potential. We also mapped a proposed cortical model of direction-selectivity that uses depressing synapses onto this chip. The results are qualitatively similar to the results obtained in the original work [Chance et al., 1998].

The similarity of the circuit responses to the responses from Abbott and colleagues's synaptic model means that we can use these VLSI networks of integrate-and-fire (I/F) neurons as an alternative to computer simulations of dynamical networks composed of large numbers of integrate-and-fire neurons using synapses with different time constants. The outputs of such networks can also be used to interface with neural wetware. An infrastructure for a re-programmable, reconfigurable, multi-chip neuronal system is being developed along with a user-defined interface so that the system is easily accessible to a naive user.

## 2 Comparisons between Models of Depression

We compare the circuit model with the theoretical model from [Abbott et al., 1997] describing synaptic depression and facilitation. Similar comparisons with [Tsodyks and Markram, 1997] give the same conclusions. Here, we only describe the circuit model for synaptic depression. The equivalent model for facilitation is described elsewhere [Liu, 2002].

### 2.1 Theoretical Model for Depression Model

In the model from [Abbott et al., 1997], the synaptic strength is described by $g\,D(t)$, where $D$ is a variable between 0 and 1 that describes the amount of depression ($D = 1$ means no depression) and $g$ is the maximum synaptic strength. The recovery dynamics of $D$ is:

$$\tau_d\,\frac{dD}{dt} = 1 - D \tag{1}$$

where $\tau_d$ is the recovery time of the depression. The update equation for $D$ right after a spike at time $t = t_{sp}$ is

$$D(t_{sp}^{+}) = d\,D(t_{sp}^{-}) \tag{2}$$

where $d$ ($d < 1$) is the amount by which $D$ is decreased right after the spike and $t_{sp}$ is the time of the spike. The average steady-state value of depression for a regular spike train with a rate $r$ is

$$D_{ss} = \frac{1 - e^{-1/(r\,\tau_d)}}{1 - d\,e^{-1/(r\,\tau_d)}}. \tag{3}$$

### 2.2 Circuit Model of Depressing Synapse

In this circuit model of synaptic depression, the equation that describes the recovery dynamics of the depressing variable, $D$ is nonlinear. This nonlinearity comes about because the exponential dynamics in Eq. 1 was replaced with the dynamics of the current through a single diode-connected transistor. Hence, the equation describing the recovery of $D$ (derived from the circuit in the region where a transistor operates in the subthreshold region or the current is exponential in the gate voltage of the transistor) can be formulated as

$$\frac{dD}{dt} = M\,\left(1 - D^{1/\kappa}\right) \tag{4}$$

where $1/M$ is the equivalent of $\tau_d$ in Eq. 1 and $\kappa$ (a transistor parameter) is less than 1. The maximum value of $D$ is 1. The update equation remains as before:

$$D(t_{sp}^{+}) = d\,D(t_{sp}^{-}). \tag{5}$$

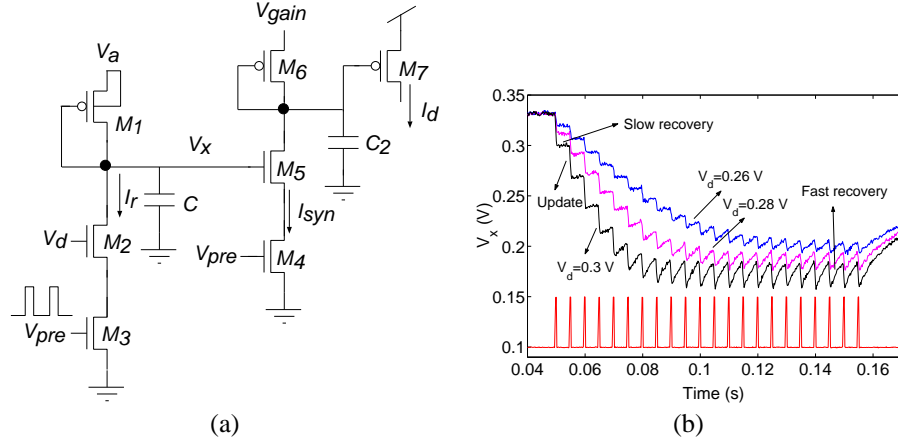

(a)                                    (b)

Figure 1: Schematic for a depressing synapse circuit and responses to a regular input spike train. (a) Depressing synapse circuit. The voltage $V_a$ determines the synaptic conductance $g$ while the synaptic term $g\,D$ or $I_{syn}$ is exponential in the voltage, $V_x$. The subcircuit consisting of transistors, $M_1$, $M_2$, and $M_3$, control the dynamics of $I_{syn}$. The presynaptic input goes to the gate terminal of $M_3$ which acts like a switch. When there is a presynaptic spike, a quantity of charge (determined by $V_d$) is removed from the node $V_x$. In between spikes, $V_x$ recovers to the voltage, $V_a$ through the diode-connected transistor, $M_1$. When there is no spike, $V_x$ is around $V_a$. When the presynaptic input comes from a regular spike train, $V_x$ decreases with each spike and recovers in between spikes. It reaches a steady-state value as shown in (b). During the spike, transistor $M_4$ turns on and the synaptic weight current $I_{syn}$ charges up the membrane potential of the neuron through the current-mirror circuit consisting of $M_6$, $M_7$, and the capacitor $C_2$. We can convert the $I_{syn}$ current source into a synaptic current $I_d$ with some gain and a "time constant" by adjusting the voltage $V_{gain}$. The decay dynamics of $I_d$ is given by $I_d(t) = \frac{I_{syn}(t=t_{sp})}{1+\frac{A\,I_{syn}(t=t_{sp})}{Q_T}}$ where $Q_T = C_2\,U_T$ and $A = e^{\kappa(V_{dd}-V_{gain})/U_T}$. In a normal synapse circuit (that is, without short-term dynamics), $V_x$ is controlled by an external bias voltage. (b) Input spike train at a frequency of 20 Hz (bottom curve) and corresponding response $V_x$ (top curve) of the circuit for $V_d = 0.26, 0.28, 0.3$ V. The diode-connected transistor $M_1$ has nonlinear dynamics. The recovery time of the depressing variable $D$ depends on the distance of the present value of $V_x$ from $V_a$. The recovery rate of $D$ increases for a larger difference between $V_x$ and $V_a$.

### 2.2.1 Circuit

Equations 4 and 5 are derived from the circuit in Fig. 1. The operation of this circuit is described in the caption. The detailed analysis leading to the differential equations for $D$ is described in [Liu, 2002]. The voltage $V_x$ codes for $g\,D$. The conductance $g$ is set by $V_a$ while the dynamics of $D$ is set by both $V_d$ and $V_a$. The time taken for the present value of $V_x$ to return to $V_a$ is determined by the current dynamics of the diode-connected transistor $M_1$ and $V_a$. The recovery time constant $(1/M)$ of $D$ is set by $V_a$.

The synaptic weight is described by the current, $I_{syn}$ in Fig. 1(a):

$$I_{syn}(t) = I_{on}e^{\kappa\,V_x/U_T} = g\,D(t) \qquad (6)$$

where $g = I_{on}e^{\kappa\,V_a/U_T}$ is the synaptic strength, $D(t)$ is $\frac{e^{\kappa(V_{dd}-V_a)/U_T}\,I_{op}}{I_{rf}(t)}$, and $I_{rf} = I_{op}\,e^{\kappa\,(V_{dd}-V_x)/U_T}$. The recovery time constant $(1/M)$ of $D$ is set by $V_a$ ($M =$

$\frac{I_{op}\,\kappa}{Q_T}\,e^{-(1-\kappa)(V_{dd}-V_a)/U_T}$). The synaptic current, $I_d$ to the neuron, is then a current source $I_{syn}$ which lasts for the duration of the pulse width of the presynaptic spike. However, we can set a longer time constant for the synaptic current through $V_{gain}$. The equation describing this dependence (that is, the current equation for a current-mirror circuit) is given in the caption of Fig. 1.

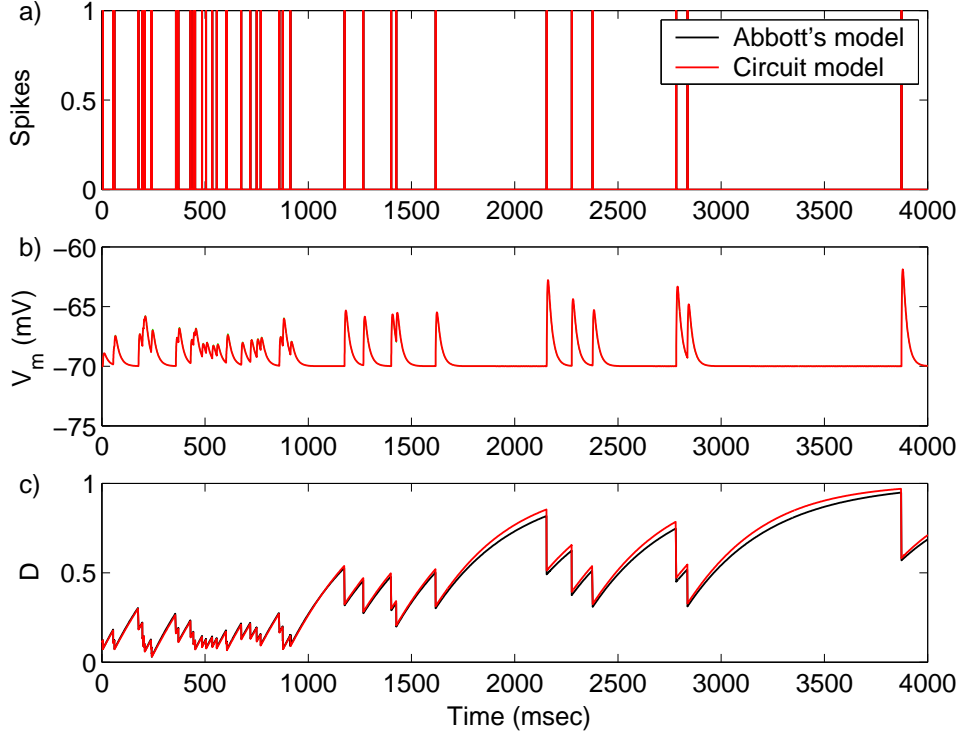

Figure 2: Comparison between the outputs of the two models of depression. An optimization algorithm was used to determine the parameters of the models so that the least square error in the difference between the EPSPs from the two models was at a minimum. The corresponding $D$ distribution is shown in (c). (a) Poisson-distributed input with an initial frequency of 40 Hz and an end frequency of 1 Hz. (b) The EPSP responses of both models were identical. (c) The $D$ values were almost identical except in the region when $D$ is close to 1. Parameters used in the simulations: $\tau_m = 20\,ms$, $d = 0.6$, $\kappa = 0.7$, $M = 2.20\,s^{-1}$.

It is difficult to compute a closed-form solution for Eq. 4 for any value of $\kappa$ (a transistor parameter which is less than 1). This value also changes under different operating conditions and between transistors fabricated in different processes. Hence, we solve for $D(t)$ in the case of $\kappa = 0.5$ given that the last spike occurred at $t = t_0$:

$$dD/dt = M(1 - D^2) \Rightarrow D(t) = \frac{D(t_0)\,\cosh\,(M\,t) + \sinh\,(M\,t)}{\cosh\,(M\,t) + D(t_0)\,\sinh\,(M\,t)}. \tag{7}$$

When $D$ is far from its recovered value of 1, we can approximate its recovery dynamics by $dD/dt = M$ (irrespective of $\kappa$) and solving for $D(t)$, we get

$$D(t) = M\,t + D(t_0).$$

In this regime, $D(t)$ follows a linear trajectory. Note that the same is true of Eq. 1 when $t << \tau_d$.

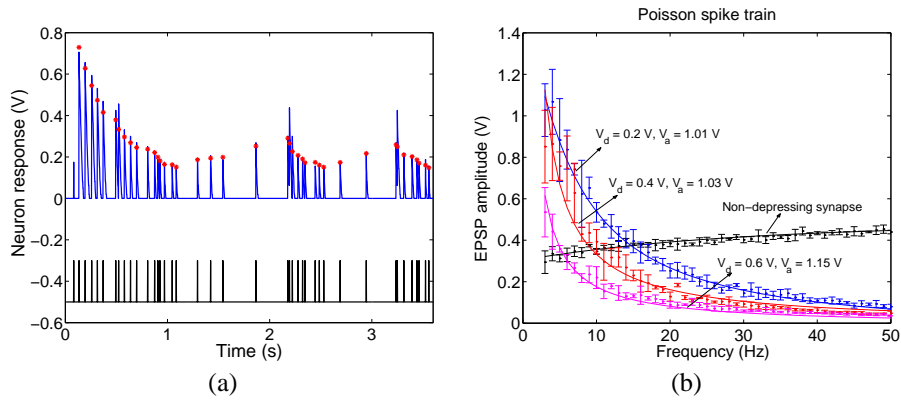

Figure 3: Transient EPSP responses to a 10 Hz Poisson-distributed train (a) and dependence of steady-state EPSP responses on the input frequency for different values of depression (b). The data was measured from the fabricated circuit. In (a), the amplitude of the EPSP decreases with each incoming input spike clearly showing the effect of synaptic depression. In (a), the EPSP amplitude depends on the occurrence of the previous spike. The asterisks are the fits of the circuit model to the peak value of each EPSP. The fits give a $d$ value of 0.79. The input is the bottom curve of the plot. (b) Steady-state EPSP amplitude versus frequency for a Poisson-distributed input. The solid lines are fits from the theoretical equation.

## 3 Comparison between Models

We compare the two models by looking at how $D$ changes in response to a Poisson-distributed input whose frequency varied from 40 Hz to 1 Hz as shown in Fig. 2. We used a simple linear differential equation to describe the dynamics of the membrane potential $V_m$:

$$\tau_m \frac{dV_m(t)}{dt} = Ri(t) - V_m(t)$$

where $\tau_m$ is the membrane time constant and $i(t)$ is the synaptic current. We ran an optimization algorithm on the parameters in the two models so that the least square error between the EPSP outputs of both models was at a minimum. In this case, the EPSP responses were identical (Fig. 2(b)) and the corresponding $D$ values (Fig. 2(c)) were almost identical except in the region where $D$ was close to the maximum value. We performed the same comparison with Tsodyks and Markram's model and the results were similar. Hence, the circuit model can be used to describe short-term synaptic depression in a network simulation. However, the nonlinear recovery dynamics of the circuit model leads a different functional dependence of the average steady-state EPSP on the frequency of a regular input spike train.

## 4 Circuit Response

The data in the figures in the remainder of this paper are obtained from a fabricated silicon network of aVLSI integrate-and-fire neurons of the type described in [Boahen, 1997, Van Schaik, 2001, Indiveri, 2000, Liu et al., 2001] with different types of synapses.

### 4.1 Transient Response

We first measured the transient response of the neuron when stimulated by a 10 Hz Poisson-distributed input through the depressing synapse. We tuned the parameters of the synapse and the leak current so that the membrane potential did not build up to threshold. This data is shown in Fig. 3(a). The fit (marked with asterisks with in the figure) using Eq. 6 along with $D$ computed from Eq. 7, describes the experimental data well.

### 4.2 Steady-State Response

The equation describing the dependence of the steady-state values of $D$ on the presynaptic frequency can easily be determined in the case of a regular spiking input of rate $r$ by using Eqs. 5 and 7. The resulting expression is somewhat complicated but by using the reduced dynamics expression $(dD/dt = M)$, we obtain a simpler expression for $D_{ss}$:

$$D_{ss} = \frac{M}{(1-d)r}. \tag{8}$$

This equation shows that the steady-state $D$ and hence, the steady-state EPSP amplitude is inversely dependent on the presynaptic rate $r$. The form of the curve is similar to the results obtained in the work of [Abbott et al., 1997] where the data can be fitted with Eq. 3.

From the chip, we measured the steady-state EPSP amplitudes using a Poisson-distributed train whose frequency varied over a range of 3 Hz to 50 Hz in steps of 1 Hz. Each frequency interval lasted 15 s and the EPSP amplitude was averaged in the last 5 s to obtain the steady-state value. Four separate trials were performed and the resulting mean and the variance of the measurements are shown in Fig. 3(b). The parameters from the fits using the response data to a regular spiking input were used to generate the fitted curve to the data in Fig. 3(b). The values from the fits give recovery time constants from 1–3 s and $D_{ss}$ values varying between 0.02-0.04.

## 5 Role of Synaptic Depression

Different computational roles have been proposed for networks which incorporate synaptic depression. In this section, we describe some measurements which illustrate the postulated roles of depression. The direction-selective model of [Chance et al., 1998] which makes use of the phase advance property from depressing synapses have been attempted on a neuron on our chip and the direction-selective results were qualitatively similar.

Depressing synapses have also been implicated in cortical gain control [Abbott et al., 1997]. A depressing synapse acts like a transient detector to changes in frequency (or a first derivative filter). A synapse with short-term depression responds equally to equal percentage rate changes in its input on different firing rates. We demonstrate the gain-control mechanism of short-term depression by measuring the neuron's response to step changes in input frequency from 10 Hz to 20 Hz to 40 Hz. Each step change represents the same rate change in input frequency. These results are shown in Fig. 4(a) for a regular train and in (b) for a Poisson-distributed train. Each frequency epoch lasted 3 s so the synaptic strength should have reached steady-state before the next increase in input frequency.

For both figures in Fig. 4, the top curve shows the response of the neuron when stimulated by the input (bottom curve) through a depressing synapse (top curve) and a non-depressing synapse (middle curve). Figure 4(a) shows clearly that the transient increase in the firing rate of a neuron when stimulated through a depressing synapse right after each step increase in input frequency and the subsequent adaptation of its firing rate to a steady-state value. The steady-state firing rate of the neuron with a depressing synapse is less dependent on the

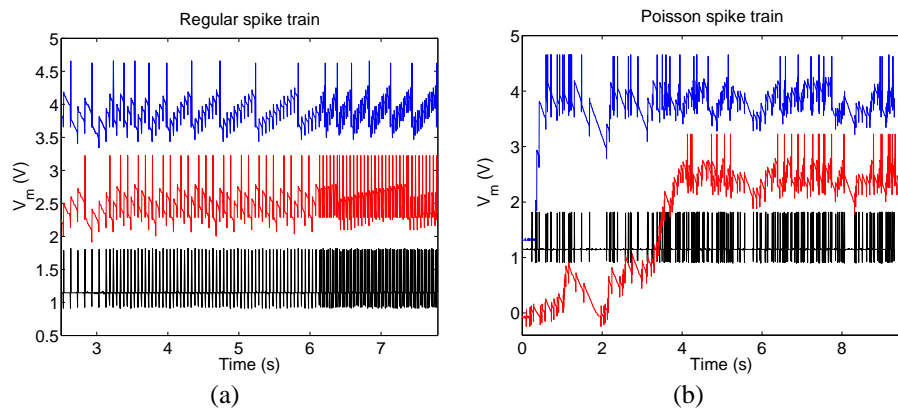

Figure 4: Response of neuron to changes in input frequency (bottom curve) when stimulated through a depressing synapse (top curve) and a non-depressing synapse (middle curve). The neuron was stimulated for three frequency intervals (10 Hz to 20 Hz to 40 Hz) lasting 3 s each. (a) Response of neuron using a regular spiking input. The steady-state firing rate of the neuron increased almost linearly with the input frequency when stimulated through the non-depressing synapse. In the depressing-synapse curve, there is a transient increase in the neuron's firing rate before the rate adapted to steady-state. (b) Response of neuron using a Poisson-distributed input. The parameters for both types of synapses were tuned so that the steady-state firing rates were about the same at the end of each frequency interval for both synapses. Notice that during the 10 Hz interval, the neuron quickly built up to threshold if it was stimulated through the depressing synapse.

absolute input frequency when compared to the firing rate of the neuron when stimulated through the non-depressing synapse. In the latter case, the firing rate of the neuron is approximately linear in the input rate.

The data in Fig. 4(b) obtained from a Poisson-distributed train shows an obvious difference in the responses between the depressing and non-depressing synapse. In the depressing-synapse case, the neuron quickly reached threshold for a 10 Hz input, while it remained sub-threshold in the non-depressing case until the input has increased to 20 Hz. This suggests that a potential role of a depressing synapse is to drive a neuron quickly to threshold when its membrane potential is far away from its threshold.

# 6   Conclusion

We described a model of synaptic depression that was derived from a circuit implementation. This circuit model has nonlinear recovery dynamics in contrast to current theoretical models of dynamic synapses. It gives qualitatively similar results when compared to the model of Abbott and colleagues. Measured data from a chip with aVLSI integrate-and-fire neurons and dynamic synapses show that this network can be used to simulate the responses of dynamic networks with short-term dynamic synapses. Experimental results suggest that depressing synapses can be used to drive a neuron quickly up to threshold if its membrane potential is at the resting potential. The silicon networks provide an alternative to computer simulation of spike-based processing models with different time constant synapses because they run in real-time and the computational time does not scale with the size of the neuronal network.

**Acknowledgments**

This work was supported in part by the Swiss National Foundation Research SPP grant. We acknowledge Kevan Martin, Pamela Baker, and Ora Ohana for many discussions on dynamic synapses.

# References

[Abbott et al., 1997] Abbott, L., Sen, K., Varela, J., and Nelson, S. (1997). Synaptic depression and cortical gain control. *Science*, 275(5297):220–223.

[Boahen, 1997] Boahen, K. A. (1997). *Retinomorphic Vision Systems: Reverse Engineering the Vertebrate Retina*. PhD thesis, California Institute of Technology, Pasadena CA.

[Chance et al., 1998] Chance, F., Nelson, S., and Abbott, L. (1998). Synaptic depression and the temporal response characteristics of V1 cells. *Journal of Neuroscience*, 18(12):4785–4799.

[Indiveri, 2000] Indiveri, G. (2000). Modeling selective attention using a neuromorphic aVLSI device. *Neural Computation*, 12(12):2857–2880.

[Liu, 2002] Liu, S.-C. (2002). Dynamic synapses and neuron circuits for mixed-signal processing. *EURASIP Journal on Applied Signal Processing: Special Issue*. Submitted.

[Liu et al., 2001] Liu, S.-C., Kramer, J., Indiveri, G., Delbrück, T., Burg, T., and Douglas, R. (2001). Orientation-selective aVLSI spiking neurons. *Neural Networks: Special Issue on Spiking Neurons in Neuroscience and Technology*, 14(6/7):629–643.

[Maass and Zador, 1999] Maass, W. and Zador, A. (1999). Computing and learning with dynamic synapses. In Maass, W. and Bishop, C. M., editors, *Pulsed Neural Networks*, chapter 6, pages 157–178. MIT Press, Boston, MA. ISBN 0-262-13350-4.

[Matveev and Wang, 2000] Matveev, V. and Wang, X. (2000). Differential short-term synaptic plasticity and transmission of complex spike trains: to depress or to facilitate? *Cerebral Cortex*, 10(11):1143–1153.

[Rasche and Hahnloser, 2001] Rasche, C. and Hahnloser, R. (2001). Silicon synaptic depression. *Biological Cybernetics*, 84(1):57–62.

[Senn et al., 1998] Senn, W., Segev, I., and Tsodyks, M. (1998). Reading neuronal synchrony with depressing synapses. *Neural Computation*, 10(4):815–819.

[Stratford et al., 1998] Stratford, K., Tarczy-Hornoch, K., Martin, K., Bannister, N., and Jack, J. (1998). Excitatory synaptic inputs to spiny stellate cells in cat visual cortex. *Nature*, 382:258–261.

[Tsodyks and Markram, 1997] Tsodyks, M. and Markram, H. (1997). The neural code between neocortical pyramidal neurons depends on neurotransmitter release probability. *Proc. Natl. Acad. Sci. USA*, 94(2).

[Tsodyks et al., 1998] Tsodyks, M., Pawelzik, K., and Markram, H. (1998). Neural networks with dynamic synapses. *Neural Computation*, 10(4):821–835.

[Van Schaik, 2001] Van Schaik, A. (2001). Building blocks for electronic spiking neural networks. *Neural Networks*, 14(6/7):617–628. Special Issue on Spiking Neurons in Neuroscience and Technology.

[Varela et al., 1997] Varela, J., Sen, K., Gibson, J., Fost, J., Abbott, L., and Nelson, S. (1997). A quantitative description of short-term plasticity at excitatory synapses in layer 2/3 of rat primary visual cortex. *Journal of Neuroscience*, 17(20):7926–7940.
